# Nonlinear Blind Source Separation by Integrating Independent Component Analysis and Slow Feature Analysis

**Tobias Blaschke**
Institute for Theoretical Biology
Humboldt University Berlin
Invalidenstraße 43, D-10115 Berlin, Germany
t.blaschke@biologie.hu-berlin.de

**Laurenz Wiskott**
Institute for Theoretical Biology
Humboldt University Berlin
Invalidenstraße 43, D-10115 Berlin, Germany
l.wiskott@biologie.hu-berlin.de

## Abstract

In contrast to the equivalence of linear blind source separation and linear independent component analysis it is not possible to recover the original source signal from some unknown nonlinear transformations of the sources using only the independence assumption. Integrating the objectives of statistical independence and temporal slowness removes this indeterminacy leading to a new method for nonlinear blind source separation. The principle of temporal slowness is adopted from slow feature analysis, an unsupervised method to extract slowly varying features from a given observed vectorial signal. The performance of the algorithm is demonstrated on nonlinearly mixed speech data.

## 1   Introduction

Unlike in the linear case the nonlinear Blind Source Separation (BSS) problem can not be solved solely based on the principle of statistical independence [1, 2]. Performing nonlinear BSS with Independent Component Analysis (ICA) requires additional information about the underlying sources or to regularize the nonlinearities. Since source signal components are usually more slowly varying than any nonlinear mixture of them we consider to require the estimated sources to be as slowly varying as possible. This can be achieved by incorporating ideas from Slow Feature Analysis (SFA) [3] into ICA.

After a short introduction to linear BSS, nonlinear BSS, and SFA we will show a way how to combine SFA and ICA to obtain an algorithm that solves the nonlinear BSS problem.

## 2  Linear Blind Source Separation

Let $\mathbf{x}(t) = [x_1(t), \ldots, x_N(t)]^T$ be a linear mixture of a source signal $\mathbf{s}(t) = [s_1(t), \ldots, s_N(t)]^T$ and defined by

$$\mathbf{x}(t) = \mathbf{A}\mathbf{s}(t), \tag{1}$$

with an invertible $N \times N$ mixing matrix $\mathbf{A}$. Finding a mapping

$$\mathbf{u}(t) = \mathbf{Q}\mathbf{W}\mathbf{x}(t) \tag{2}$$

such that the components of $\mathbf{u}$ are mutually statistically independent is called Independent Component Analysis (ICA). The mapping is often divided into a whitening mapping $\mathbf{W}$, resulting in uncorrelated signal components $y_i$ with unit variance and a successive orthogonal transformation $\mathbf{Q}$, because one can show [4] that after whitening an orthogonal transformation is sufficient to obtain independence. It is well known that ICA solves the linear BSS problem [4]. There exists a variety of algorithms performing ICA and therefore BSS (see e.g. [5, 6, 7]). Here we focus on a method using only second-order statistics introduced by Molgedey and Schuster [8]. The method consists of optimizing an objective function subject to minimization, which can be written as

$$\Psi_{\text{ICA}}(\mathbf{Q}) = \sum_{\substack{\alpha,\beta=1 \\ \alpha \neq \beta}}^{N} \left( C_{\alpha\beta}^{(\mathbf{u})}(\tau) \right)^2 = \sum_{\substack{\alpha,\beta=1 \\ \alpha \neq \beta}}^{N} \left( \sum_{\gamma,\delta=1}^{N} Q_{\alpha\gamma} Q_{\beta\delta} C_{\gamma\delta}^{(\mathbf{y})}(\tau) \right)^2, \tag{3}$$

operating on the already whitened signal $\mathbf{y}$. $C_{\gamma\delta}^{(\mathbf{y})}(\tau)$ is an entry of a symmetrized time delayed covariance matrix defined by

$$\mathbf{C}^{(\mathbf{y})}(\tau) = \left\langle \mathbf{y}(t)\mathbf{y}(t+\tau)^T + \mathbf{y}(t+\tau)\mathbf{y}(t)^T \right\rangle, \tag{4}$$

and $\mathbf{C}^{(\mathbf{u})}(\tau)$ is defined correspondingly. $Q_{\alpha\beta}$ denotes an entry of $\mathbf{Q}$. Minimization of $\Psi_{\text{ICA}}$ can be understood intuitively as finding an orthogonal matrix $\mathbf{Q}$ that diagonalizes the covariance matrix with time delay $\tau$. Since, because of the whitening, the instantaneous covariance matrix is already diagonal this results in signal components that are decorrelated instantaneously and at a given time delay $\tau$. This can be sufficient to achieve statistical independence [9].

### 2.1  Nonlinear BSS and ICA

An obvious extension to the linear mixing model (1) has the form

$$\mathbf{x}(t) = F(\mathbf{s}(t)), \tag{5}$$

with a function $F(\cdot)\ \mathbb{R}^N \to \mathbb{R}^M$ that maps $N$-dimensional source vectors $\mathbf{s}$ onto $M$-dimensional signal vectors $\mathbf{x}$. The components $x_i$ of the observable are a nonlinear mixture of the sources and like in the linear case source signal components $s_i$ are assumed to be mutually statistically independent. Extracting the source signal is in general only possible if $F(\cdot)$ is an invertible function, which we will assume from now on.

The equivalence of BSS and ICA in the linear case does in general not hold for a nonlinear function $F(\cdot)$ [1, 2]. To solve the nonlinear BSS problem additional constraints on the mixture or the estimated signals are needed to bridge the gap between ICA and BSS. Here we propose a new way to achieve this by adding a slowness objective to the independence objective of pure ICA. Assume for example a sinusoidal signal component $x_i = \sin(2\pi t)$ and a second component that is the square of the first $x_j = x_i^2 = 0.5(1 - \cos(4\pi t))$ is given. The second component is more quickly varying due to the frequency doubling

induced by the squaring. Typically nonlinear mixtures of signal components are more quickly varying than the original components. To extract the right source components one should therefore prefer the slowly varying ones. The concept of slowness is used in our approach to nonlinear BSS by combining an ICA part that provides the independence of the estimated source signal components with a part that prefers slowly varying signals over more quickly varying ones. In the next section we will give a short introduction to Slow Feature Analysis (SFA) building the basis of the second part of our method.

## 3 Slow Feature Analysis

Assume a vectorial input signal $\mathbf{x}(t) = [x_1(t), \ldots, x_M(t)]^T$ is given. The objective of SFA is to find an in general nonlinear input-output function $\mathbf{u}(t) = \mathbf{g}(\mathbf{x}(t))$ with $\mathbf{g}(\mathbf{x}(t)) = [g_1(\mathbf{x}(t)), \ldots, g_R(\mathbf{x}(t))]^T$ such that the $u_i(t)$ are varying as slowly as possible. This can be achieved by successively minimizing the objective function

$$\Delta(u_i) \quad := \quad \langle \dot{u}_i^2 \rangle \tag{6}$$

for each $u_i$ under the constraints

$$\langle u_i \rangle \quad = \quad 0 \qquad \qquad \text{(zero mean)}, \tag{7}$$
$$\langle u_i^2 \rangle \quad = \quad 1 \qquad \qquad \text{(unit variance)}, \tag{8}$$
$$\langle u_i u_j \rangle \quad = \quad 0 \;\; \forall \, j < i \qquad \text{(decorrelation and order)}. \tag{9}$$

Constraints (7) and (8) ensure that the solution will not be the trivial solution $u_i = \text{const.}$ Constraint (9) provides uncorrelated output signal components and thus guarantees that different components carry different information. Intuitively we are searching for signal components $u_i$ that have on average a small slope.

Interestingly Slow Feature Analysis (SFA) can be reformulated with an objective function similar to second-order ICA, subject to maximization [10],

$$\Psi_{\text{SFA}}(\mathbf{Q}) \quad = \quad \sum_{\alpha=1}^{M} \left( C_{\alpha\alpha}^{(\mathbf{u})}(\tau) \right)^2 = \sum_{\alpha=1}^{M} \left( \sum_{\beta,\gamma=1}^{M} Q_{\alpha\beta} Q_{\alpha\gamma} C_{\beta\gamma}^{(\mathbf{y})}(\tau) \right)^2 . \tag{10}$$

To understand (10) intuitively we notice that slowly varying signal components are easier to predict, and should therefore have strong auto correlations in time. Thus, maximizing the time delayed variances produces slowly varying signal components.

## 4 Independent Slow Feature Analysis

If we combine ICA and SFA we obtain a method we refer to as Independent Slow Feature Analysis (ISFA) that recovers independent components out of a nonlinear mixture using a combination of SFA and second-order ICA. As already explained, second-order ICA tends to make the output components independent and SFA tends to make them slow. Since we are dealing with a nonlinear mixture we first compute a nonlinearly expanded signal $\mathbf{z} = \mathbf{h}(\mathbf{x})$ with $\mathbf{h}(\cdot) \, \mathbb{R}^M \rightarrow \mathbb{R}^L$ being typically monomials up to a given degree, e.g. an expansion with monomials up to second degree can be written as

$$\mathbf{h}(\mathbf{x}(t)) = [x_1, \, \ldots, \, x_N, \, x_1 x_1, \, x_1 x_2, \, \ldots, \, x_M x_M]^T - \mathbf{h}_0^T \tag{11}$$

when given an $M$-dimensional signal $\mathbf{x}$. The constant vector $\mathbf{h}_0^T$ is used to make the expanded signal mean free. In a second step $\mathbf{z}$ is whitened to obtain $\mathbf{y} = \mathbf{W}\mathbf{z}$. Thirdly we apply linear ICA combined with linear SFA on $\mathbf{y}$ in order to find the estimated source signal

**u**. Because of the whitening we know that ISFA, like ICA and SFA, is solved by finding an orthogonal $L \times L$ matrix $\mathbf{Q}$. We write the estimated source signal **u** as

$$\mathbf{v} = \left( \begin{array}{c} \mathbf{u} \\ \tilde{\mathbf{u}} \end{array} \right) = \mathbf{Q}\mathbf{y} = \mathbf{Q}\mathbf{W}\mathbf{z} = \mathbf{Q}\mathbf{W}\mathbf{h}\left(\mathbf{x}\right) , \tag{12}$$

where we introduced $\tilde{\mathbf{u}}$, since $R$, the dimension of the estimated source signal $\mathbf{u}$, is usually much smaller than $L$, the dimension of the expanded signal. While the $u_i$ are statistically independent and slowly varying the components $\tilde{u}_i$ are more quickly varying and may be statistically dependent on each other as well as on the selected components.

To summarize, we have an $M$ dimensional input **x** an $L$ dimensional nonlinearly expanded and whitened **y** and an $R$ dimensional estimated source signal **u**. ISFA searches an $R$ dimensional subspace such that the $u_i$ are independent and slowly varying. This is achieved at the expense of all $\tilde{u}_i$.

### 4.1 Objective function

To recover $R$ source signal components $u_i$ $i = 1, \ldots, R$ out of an $L$-dimensional expanded and whitened signal **y** the objective reads

$$\Psi_{\text{ISFA}}\left(u_1, \ldots, u_R; \tau\right) = b_{\text{ICA}} \sum_{\substack{\alpha,\beta=1, \\ \alpha \neq \beta}}^{R} \left( C_{\alpha\beta}^{(\mathbf{u})}\left(\tau\right) \right)^2 - b_{\text{SFA}} \sum_{\alpha=1}^{R} \left( C_{\alpha\alpha}^{(\mathbf{u})}\left(\tau\right) \right)^2 , \tag{13}$$

where we simply combine the ICA objective (3) and SFA objective (10) weighted by the factors $b_{\text{ICA}}$ and $b_{\text{SFA}}$, respectively. Note that the ICA objective is usually applied to the linear case to unmix the linear whitened mixture **y** whereas here it is used on the nonlinearly expanded whitened signal $\mathbf{y} = \mathbf{W}\mathbf{z}$. ISFA tries to minimize $\Psi_{\text{ISFA}}$ which is the reason why the SFA part has a negative sign.

### 4.2 Optimization Procedure

From (12) we know that $\mathbf{C}^{(\mathbf{u})}\left(\tau\right)$ in (13) depends on the orthogonal matrix $\mathbf{Q}$. There are several ways to find the orthogonal matrix that minimizes the objective function. Here we apply successive Givens rotations to obtain $\mathbf{Q}$. A Givens rotation $\mathbf{Q}^{\mu\nu}$ is a rotation around the origin within the plane of two selected components $\mu$ and $\nu$ and has the matrix form

$$Q_{\alpha\beta}^{\mu\nu} := \begin{cases} \cos(\phi) & \text{for } (\alpha,\beta) \in \{(\mu,\mu),(\nu,\nu)\} \\ -\sin(\phi) & \text{for } (\alpha,\beta) \in \{(\mu,\nu)\} \\ \sin(\phi) & \text{for } (\alpha,\beta) \in \{(\nu,\mu)\} \\ \delta_{\alpha\beta} & \text{otherwise} \end{cases} \tag{14}$$

with Kronecker symbol $\delta_{\alpha\beta}$ and rotation angle $\phi$. Any orthogonal $L \times L$ matrix such as $\mathbf{Q}$ can be written as a product of $\frac{L(L-1)}{2}$ (or more) Givens rotation matrices $\mathbf{Q}^{\mu\nu}$ (for the rotation part) and a diagonal matrix with elements $\pm 1$ (for the reflection part). Since reflections do not matter in our case we only consider the Givens rotations as is often used in second-order ICA algorithms (see e.g. [11]).

We can therefore write the objective as a function of a Givens rotation $\mathbf{Q}^{\mu\nu}$ as

$$\begin{aligned} \Psi_{\text{ISFA}}\left(\mathbf{Q}^{\mu\nu}\right) = \; & b_{\text{ICA}} \sum_{\substack{\alpha,\beta=1, \\ \alpha \neq \beta}}^{R} \left( \sum_{\gamma,\delta=1}^{L} Q_{\alpha\gamma}^{\mu\nu} Q_{\beta\delta}^{\mu\nu} C_{\gamma\delta}^{(\mathbf{y})}\left(\tau\right) \right)^2 - \\ & b_{\text{SFA}} \sum_{\alpha=1}^{R} \left( \sum_{\beta,\gamma=1}^{L} Q_{\alpha\beta}^{\mu\nu} Q_{\alpha\gamma}^{\mu\nu} C_{\beta\gamma}^{(\mathbf{y})}\left(\tau\right) \right)^2 . \end{aligned} \tag{15}$$

Assume we want to minimize $\Psi_{\text{ISFA}}$ for a given $R$, where $R$ denotes the number of signal components we want to extract. Applying a Givens rotation $\mathbf{Q}^{\mu\nu}$ we have to distinguish three cases.

- **Case 1:** Both axes $u_\mu$ and $u_\nu$ lie inside the subspace spanned by the first $R$ axes $(\mu, \nu \leq R)$. The sum over all squared cross correlations of all signal components that lie outside the subspace is constant as well as those of all signal components inside the subspace. There is no interaction between inside and outside, in fact the objective function is exactly the objective for an ICA algorithm based on second-order statistics e.g. TDSEP or SOBI [12, 13]. In [10] it has been shown that this is equivalent to SFA in the case of a single time delay.

- **Case 2:** Only one axis, w.l.o.g. $u_\mu$, lies inside the subspace, the other, $u_\nu$, outside $(\mu \leq R < \nu)$. Since one axis of the rotation plane lies outside the subspace, $u_\mu$ in the objective function can be optimized at the expense of $\tilde{u}_\nu$ outside the subspace. A rotation of $\pi/2$, for instance, would simply exchange components $u_\mu$ and $u_\nu$. This gives the possibility to find the slowest and most independent components in the whole space spanned by all $u_i$ and $\tilde{u}_j$ $(i = 1, \ldots, R, \; j = R+1, \ldots, L)$ in contrast to Case 1 where the minimum is searched within the subspace spanned by the $R$ components in the objective function.

- **Case 3:** Both axes lie outside the subspace $(R < \mu, \nu)$: A Givens rotation with the two rotation axes outside the relevant subspace does not affect the objective function and can therefore be disregarded.

It can be shown that like in [14] the objective function (15) as a function of $\phi$ can always be written in the form

$$\Psi_{\text{ISFA}}^{\mu\nu}(\phi) = A_0 + A_2 \cos(2\phi + \phi_2) + A_4 \cos(4\phi + \phi_4), \qquad (16)$$

where the second term on the right hand side vanishes for Case 1. There exists a single minimum (if w.l.o.g. $\phi \in \left[-\frac{\pi}{2}, \frac{\pi}{2}\right]$) that can easily be calculated (see e.g.[14]). The derivation of (16) involves various trigonometric identities and, because of its length, is documented elsewhere[1].

It is important to notice that the rotation planes of the Givens rotations are selected from the whole $L$-dimensional space whereas the objective function only uses information of correlations among the first $R$ signal components $u_i$. Successive application of Givens rotations $\mathbf{Q}^{\mu\nu}$ leads to the final rotation matrix $\mathbf{Q}$ which is in the ideal case such that $\mathbf{Q}^T \mathbf{C}^{(\mathbf{y})}(\tau) \mathbf{Q} = \mathbf{C}^{(\mathbf{v})}(\tau)$ has a diagonal $R \times R$ submatrix $\mathbf{C}^{(\mathbf{u})}(\tau)$, but it is not clear if the final minimum is also the global one. However, in various simulations no local minima have been found.

### 4.3 Incremental Extracting of Independent Components

It is possible to find the number of independent source signal components $R$ by successively increasing the number of components to be extracted. In each step the objective function (13) is optimized for a fixed $R$. First a single signal component is extracted $(R = 1)$ and then an additional one $(R = 2)$ etc. The algorithm is stopped when no additional signal component can be extracted. As a stopping criterion every suitable measure of independence can be applied; we used the sum over squared cross-cumulants of fourth order. In our artificial examples this value is typically small for independent components, and increases by two orders of magnitudes if the number of components to be extracted is greater than the number of original source signal components.

## 5    Simulation

Here we show a simple example, with two nonlinearly mixed signal components as shown in Figure 1. The mixture is defined by

$$
\begin{aligned}
x_1(t) &= (s_1(t)+1)\sin(\pi s_2(t))\,, \\
x_2(t) &= (s_1(t)+1)\cos(\pi s_2(t))\,.
\end{aligned}
\tag{17}
$$

We used the ISFA algorithm with different nonlinearities (see Tab. 1). Already a nonlinear expansion with monomials up to degree three was sufficient to give good results in extracting the original source signal (see Fig. 1). In all cases ISFA did find exactly two independent signal components. A linear BSS method failed completely to find a good unmixing matrix.

## 6    Conclusion

We have shown that connecting the ideas of slow feature analysis and independent component analysis into ISFA is a possible way to solve the nonlinear blind source separation problem. SFA enforces the independent components of ICA to be slowly varying which seems to be a good way to discriminate between the original and nonlinearly distorted source signal components. A simple simulation showed that ISFA is able to extract the original source signal out of a nonlinear mixture. Furthermore ISFA can predict the number of source signal components via an incremental optimization scheme.

**Acknowledgments**

This work has been supported by the Volkswagen Foundation through a grant to LW for a junior research group.

**References**

[1] A. Hyvärinen and P. Pajunen. Nonlinear independent component analysis: existence and uniqueness results. *Neural Networks*, 12(3):429–439, 1999.

[2] C. Jutten and J. Karhunen. Advances in nonlinear blind source separation. In *Proc. of the 4th Int. Symposium on Independent Component Analysis and Blind Signal Separation, Nara, Japan, (ICA 2003)*, pages 245–256, 2003.

[3] Laurenz Wiskott and Terrence Sejnowski. Slow feature analysis: Unsupervised learning of invariances. *Neural Computation*, 14(4):715–770, 2002.

Table 1: Correlation coefficients of extracted ($u_1$ and $u_2$) and original ($s_1$ and $s_2$) source signal components

|  | linear | | degree 2 | | degree 3 | | degree 4 | |
|---|---|---|---|---|---|---|---|---|
|  | $u_1$ | $u_2$ | $u_1$ | $u_2$ | $u_1$ | $u_2$ | $u_1$ | $u_2$ |
| $s_1$ | -0.803 | -0.544 | -0.001 | -0.978 | 0.001 | 0.995 | 0.002 | 0.995 |
| $s_2$ | 0.332 | 0.517 | -0.988 | -0.001 | -0.995 | 0.001 | -0.996 | 0.000 |

Correlation coefficients of extracted ($u_1$ and $u_2$) and original ($s_1$ and $s_2$) source signal components for linear ICA (first column) and ISFA with different nonlinearities (monomials up to degree 2, 3, and 4). Using monomials up to degree 3 in the nonlinear expansion step already suffices to extract the original source signal. Note that the source signal can only be estimated up to permutation and scaling, resulting in different signs and permutations of the two estimated source signal components.

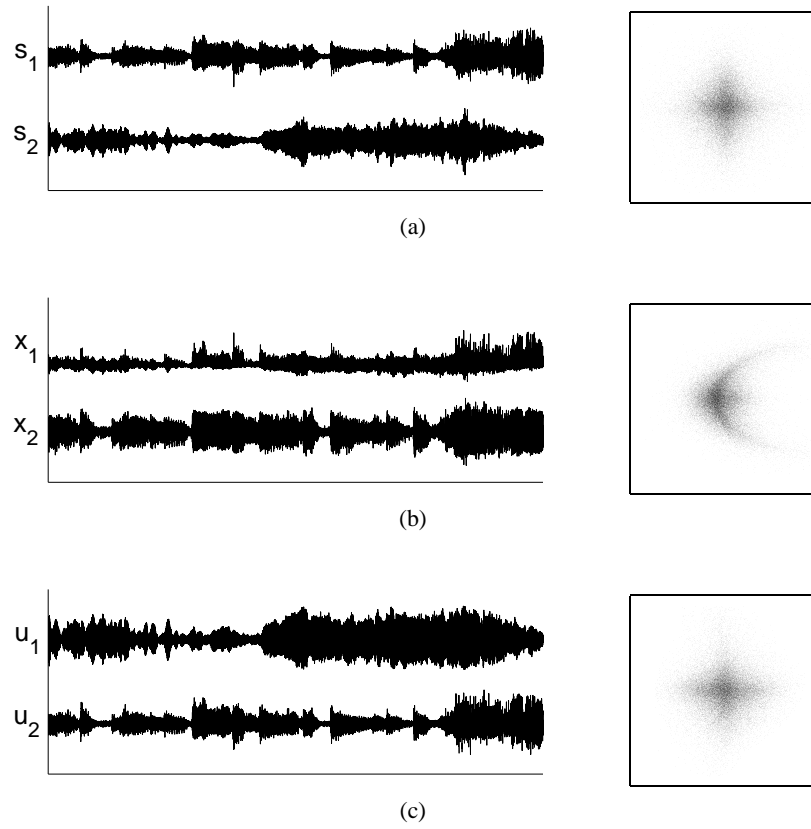

Figure 1: Waveforms and Scatter-plots of **(a)** the original source signal components $s_i$, **(b)** the nonlinear mixture, and **(c)** recovered components with nonlinear ISFA ($u_i$). As a nonlinearity we used all monomials up to degree 4.

[4] P. Comon. Independent component analysis, a new concept? *Signal Processing*, 36(3):287–314, 1994. Special Issue on Higher-Order Statistics.

[5] J.-F. Cardoso and A. Souloumiac. Blind beamforming for non Gaussian signals. *IEE Proceedings-F*, 140:362–370, 1993.

[6] T.-W. Lee, M. Girolami, and T.J. Sejnowski. Independent component analysis using an extended Infomax algorithm for mixed sub-Gaussian and super-Gaussian sources. *Neural Computation*, 11(2):409–433, 1999.

[7] A. Hyvärinen. Fast and robust fixed-point algorithms for independent component analysis. *IEEE Transactions on Neural Networks*, 10(3):626–634, 1999.

[8] L. Molgedey and G. Schuster. Separation of a mixture of independent signals using time delayed correlations. *Physical Review Letters*, 72(23):3634–3637, 1994.

[9] Lang Tong, Ruey-wen Liu, Victor C. Soon, and Yih-Fang Huang. Indeterminacy and identifiability of blind identification. *IEEE Transactions on Circuits and Systems*, 38(5):499–509, may 1991.

[10] T. Blaschke, L. Wiskott, and P. Berkes. What is the relation between independent component analysis and slow feature analysis? *(in preparation)*, 2004.

[11] Jean-François Cardoso and Antoine Souloumiac. Jacobi angles for simultaneous diagonalization. *SIAM J. Mat. Anal. Appl.*, 17(1):161–164, 1996.

[12] A. Ziehe and K.-R. Müller. TDSEP – an efficient algorithm for blind separation using time structure. In *Proc. of the 8th Int. Conference on Artificial Neural Networks (ICANN'98)*, pages 675 – 680, Berlin, 1998. Springer Verlag.

[13] Adel Belouchrani, Karim Abed Meraim, Jean-François Cardoso, and Éric Moulines. A blind source separation technique based on second order statistics. *IEEE Transactions on Signal Processing*, 45(2):434–44, 1997.

[14] T. Blaschke and L. Wiskott. CuBICA: Independent component analysis by simultaneous third- and fourth-order cumulant diagonalization. *IEEE Transactions on Signal Processing*, 52(5):1250–1256, 2004.

## Footnotes

[1] http://itb.biologie.hu-berlin.de/~blaschke
